# Minimax and Hamiltonian Dynamics of Excitatory-Inhibitory Networks

**H. S. Seung, T. J. Richardson**
Bell Labs, Lucent Technologies
Murray Hill, NJ 07974
{seung|tjr}@bell-labs.com

**J. C. Lagarias**
AT&T Labs–Research
180 Park Ave. D-130
Florham Park, NJ 07932
jcl@research.att.com

**J. J. Hopfield**
Dept. of Molecular Biology
Princeton University
Princeton, NJ 08544
jhopfield@watson.princeton.edu

## Abstract

A Lyapunov function for excitatory-inhibitory networks is constructed. The construction assumes symmetric interactions within excitatory and inhibitory populations of neurons, and antisymmetric interactions between populations. The Lyapunov function yields sufficient conditions for the global asymptotic stability of fixed points. If these conditions are violated, limit cycles may be stable. The relations of the Lyapunov function to optimization theory and classical mechanics are revealed by minimax and dissipative Hamiltonian forms of the network dynamics.

The dynamics of a neural network with symmetric interactions provably converges to fixed points under very general assumptions[1, 2]. This mathematical result helped to establish the paradigm of neural computation with fixed point attractors[3]. But in reality, interactions between neurons in the brain are asymmetric. Furthermore, the dynamical behaviors seen in the brain are not confined to fixed point attractors, but also include oscillations and complex nonperiodic behavior. These other types of dynamics can be realized by asymmetric networks, and may be useful for neural computation. For these reasons, it is important to understand the global behavior of asymmetric neural networks.

The interaction between an excitatory neuron and an inhibitory neuron is clearly asymmetric. Here we consider a class of networks that incorporates this fundamental asymmetry of the brain's microcircuitry. Networks of this class have distinct populations of excitatory and inhibitory neurons, with antisymmetric interactions

between populations and symmetric interactions within each population. Such networks display a rich repertoire of dynamical behaviors including fixed points, limit cycles[4, 5] and traveling waves[6].

After defining the class of excitatory-inhibitory networks, we introduce a Lyapunov function that establishes sufficient conditions for the global asymptotic stability of fixed points. The generality of these conditions contrasts with the restricted nature of previous convergence results, which applied only to linear networks[5], or to nonlinear networks with infinitely fast inhibition[7].

The use of the Lyapunov function is illustrated with a competitive or winner-take-all network, which consists of an excitatory population of neurons with recurrent inhibition from a single neuron[8]. For this network, the sufficient conditions for global stability of fixed points also happen to be necessary conditions. In other words, we have proved global stability over the largest possible parameter regime in which it holds, demonstrating the power of the Lyapunov function. There exists another parameter regime in which numerical simulations display limit cycle oscillations[7].

Similar convergence proofs for other excitatory-inhibitory networks may be obtained by tedious but straightforward calculations. All the necessary tools are given in the first half of the paper. But the rest of the paper explains what makes the Lyapunov function especially interesting, beyond the convergence results it yields: its role in a conceptual framework that relates excitatory-inhibitory networks to optimization theory and classical mechanics.

The connection between neural networks and optimization[3] was established by proofs that symmetric networks could find *minima* of objective functions[1, 2]. Later it was discovered that excitatory-inhibitory networks could perform the minimax computation of finding *saddle points*[9, 10, 11], though no general proof of this was given at the time. Our Lyapunov function finally supplies such a proof, and one of its components is the objective function of the network's minimax computation.

Our Lyapunov function can also be obtained by writing the dynamics of excitatory-inhibitory networks in Hamiltonian form, with extra velocity-dependent terms. If these extra terms are dissipative, then the energy of the system is nonincreasing, and is a Lyapunov function. If the extra terms are not purely dissipative, limit cycles are possible. Previous Hamiltonian formalisms for neural networks made the more restrictive assumption of purely antisymmetric interactions, and did not include the effect of dissipation[12].

This paper establishes sufficient conditions for global asymptotic stability of fixed points. The problem of finding sufficient conditions for oscillatory and chaotic behavior remains open. The perspectives of minimax and Hamiltonian dynamics may help in this task.

# 1   EXCITATORY-INHIBITORY NETWORKS

The dynamics of an excitatory-inhibitory network is defined by

$$\tau_x \dot{x} + x = f(u + Ax - By), \tag{1}$$
$$\tau_y \dot{y} + y = g(v + B^T x - Cy). \tag{2}$$

The state variables are contained in two vectors $x \in R^m$ and $y \in R^n$, which represent the activities of the excitatory and inhibitory neurons, respectively.

The symbol $f$ is used in both scalar and vector contexts. The scalar function $f : R \to R$ is monotonic nondecreasing. The vector function $f : R^m \to R^m$ is

defined by applying the scalar function $f$ to each component of a vector argument, i.e., $f(x) = (f(x_1), \ldots, f(x_m))$. The symbol $g$ is used similarly.

The symmetry of interaction within each population is imposed by the constraints $A = A^T$ and $C = C^T$. The antisymmetry of interaction between populations is manifest in the occurrence of $-B$ and $B^T$ in the equations. The terms "excitatory" and "inhibitory" are appropriate with the additional constraint that the entries of matrices $A$, $B$, and $C$ are nonnegative. Though this assumption makes sense in a neurobiological context the mathematics does not depends on it. The constant vectors $u$ and $v$ represent tonic input from external sources, or alternatively bias intrinsic to the neurons.

The time constants $\tau_x$ and $\tau_y$ set the speed of excitatory and inhibitory synapses, respectively. In the limit of infinitely fast inhibition, $\tau_y = 0$, the convergence theorems for symmetric networks are applicable[1, 2], though some effort is required in applying them to the case $C \neq 0$. If the dynamics converges for $\tau_y = 0$, then there exists some neighborhood of zero in which it still converges[7]. Our Lyapunov function goes further, as it is valid for more general $\tau_y$.

The potential for oscillatory behavior in excitatory-inhibitory networks like (1) has long been known[4, 7]. The origin of oscillations can be understood from a simple two neuron model. Suppose that neuron 1 excites neuron 2, and receives inhibition back from neuron 2. Then the effect is that neuron 1 suppresses its own activity with an effective delay that depends on the time constant of inhibition. If this delay is long enough, oscillations result. However, these oscillations will die down to a fixed point, as the inhibition tends to dampen activity in the circuit. Only if neuron 1 also excites itself can the oscillations become sustained.

Therefore, whether oscillations are damped or sustained depends on the choice of parameters. In this paper we establish sufficient conditions for the global stability of fixed points in (1). The violation of these sufficient conditions indicates parameter regimes in which there may be other types of asymptotic behavior, such as limit cycles.

## 2 LYAPUNOV FUNCTION

We will assume that $f$ and $g$ are smooth and that their inverses $f^{-1}$ and $g^{-1}$ exist. If the function $f$ is bounded above and/or below, then its inverse $f^{-1}$ is defined on the appropriate subinterval of $R$. Note that the set of $(x, y)$ lying in the range of $(f, g)$ is a positive invariant set under (1) and that its closure is a global attractor for the system.

The scalar function $F$ is defined as the antiderivative of $f$, and $\bar{F}$ as the Legendre transform $\bar{F}(x) = \max_p \{px - F(p)\}$. The derivatives of these conjugate convex functions are,

$$F'(x) = f(x) , \qquad \bar{F}'(x) = f^{-1}(x) . \tag{3}$$

The vector versions of these functions are defined componentwise, as in the definition of the vector version of $f$. The conjugate convex pair $G, \bar{G}$ is defined similarly.

The Lyapunov function requires generalizations of the standard kinetic energies $\tau_x \dot{x}^2/2$ and $\tau_y \dot{y}^2/2$. These are constructed using the functions $\Phi : R^m \times R^m \to R$ and $\Gamma : R^n \times R^n \to R$, defined by

$$\Phi(p, x) = \mathbf{1}^T F(p) - x^T p + \mathbf{1}^T \bar{F}(x) , \tag{4}$$

$$\Gamma(q, y) = \mathbf{1}^T G(q) - y^T q + \mathbf{1}^T \bar{G}(y) . \tag{5}$$

The components of the vector **1** are all ones; its dimensionality should be clear from context. The function $\Phi(p, x)$ is lower bounded by zero, and vanishes on the manifold $f(p) = x$, by the definition of the Legendre transform. Setting $p = u + Ax - By$, we obtain the generalized kinetic energy $\tau_x^{-1}\Phi(u + Ax - By, x)$, which vanishes when $\dot{x} = 0$ and is positive otherwise. It reduces to $\tau_x \dot{x}^2/2$ in the special case where $f$ is the identity function.

To construct the Lyapunov function, a multiple of the saddle function

$$S = -u^T x - \frac{1}{2}x^T A x + v^T y - \frac{1}{2}y^T C y + \mathbf{1}^T \bar{F}(x) + y^T B^T x - \mathbf{1}^T \bar{G}(y) \qquad (6)$$

is added to the kinetic energy. The reason for the name "saddle function" will be explained later. Then

$$L = \tau_x^{-1}\Phi(u + Ax - By, x) + \tau_y^{-1}\Gamma(v + B^T x - Cy, y) + rS \qquad (7)$$

is a Lyapunov function provided that it is lower bounded, nonincreasing, and $\dot{L}$ only vanishes at fixed points of the dynamics. Roughly speaking, this is enough to prove the global asymptotic stability of fixed points, although some additional technical details may be involved.

In the next section, the Lyapunov function will be applied to an example network, yielding sufficient conditions for the global asymptotic stability of fixed points. In this particular network, the sufficient conditions also happen to be necessary conditions. Therefore the Lyapunov function succeeds in delineating the largest possible parameter regime in which point attractors are globally stable. Of course, there is no guarantee of this in general, but the power of the Lyapunov function is manifest in this instance.

Before proceeding to the example network, we pause to state some general conditions for $L$ to be nonincreasing. A lengthy but straightforward calculation shows that the time derivative of $L$ is given by

$$\begin{aligned}
\dot{L} = \quad & \dot{x}^T A \dot{x} - \dot{y}^T C \dot{y} \qquad\qquad\qquad\qquad\qquad\qquad\qquad (8) \\
& -(\tau_x^{-1} + r)\dot{x}^T[f^{-1}(\tau_x \dot{x} + x) - f^{-1}(x)] \\
& -(\tau_y^{-1} - r)\dot{y}^T[g^{-1}(\tau_y \dot{y} + y) - g^{-1}(y)] \ .
\end{aligned}$$

Therefore, $L$ is nonincreasing provided that

$$\max_{a,b} \frac{(a-b)^T A(a-b)}{(a-b)^T[f^{-1}(a) - f^{-1}(b)]} \quad \leq \quad 1 + r\tau_x \ , \qquad (9)$$

$$\min_{a,b} \frac{(a-b)^T C(a-b)}{(a-b)^T[g^{-1}(a) - g^{-1}(b)]} \quad \geq \quad 1 - r\tau_y \ . \qquad (10)$$

The quotients in these inequalities are generalizations of the Rayleigh-Ritz ratios of $A$ and $C$. If $f$ and $g$ were linear, the left hand sides of these inequalities would be equal to the maximum eigenvalue of $A$ and the minimum eigenvalue of $C$.

## 3    AN EXAMPLE: COMPETITIVE NETWORK

The competitive or winner-take-all network is a classic example of an excitatory-inhibitory network[8, 7]. Its population of excitatory neurons $x_i$ receives self-feedback of strength $\alpha$ and recurrent feedback from a single inhibitory neuron $y$,

$$\tau_x \dot{x}_i + x_i \quad = \quad f(u_i + \alpha x_i - y) \ , \qquad (11)$$

$$\tau_y \dot{y} + y \quad = \quad g\left(\sum_i x_i\right) \ . \qquad (12)$$

This is a special case of (1), with $A = \alpha I$, $B = 1$, and $C = 0$.

The global inhibitory neuron mediates a competitive interaction between the excitatory neurons. If the competition is very strong, a single excitatory neuron "wins," shutting off all the rest. If the competition is weak, more than one excitatory neuron can win, usually those corresponding to the larger $u_i$. Depending on the choice of $f$ and $g$, self-feedback $\alpha$, and time scales $\tau_x$ and $\tau_y$, this network exhibits a variety of dynamical behaviors, including a single point attractor, multiple point attractors, and limit cycles[5, 7].

We will consider the specific case where $f$ and $g$ are the rectification nonlinearity $[x]^+ \coloneqq \max\{x, 0\}$. The behavior of this network will be described in detail elsewhere; only a brief summary is given here. With either of two convenient choices for $r$, $r = \tau_y^{-1}$ or $r = \alpha - \tau_x^{-1}$, it can be shown that the resulting $L$ is bounded below for $\alpha < 2$ and nonincreasing for $\alpha < \tau_x^{-1} + \tau_y^{-1}$. These are sufficient conditions for the global stability of fixed points. They also turn out to be necessary conditions, as it can be verified that the fixed points are locally unstable if the conditions are violated. The behaviors in the parameter regime defined by these conditions can be divided into two rough categories. For $\alpha < 1$, there is a unique point attractor, at which more than one excitatory neuron can be active, in a soft form of winner-take-all. For $\alpha > 1$, more than one point attractor may exist. Only one excitatory neuron is active at each of these fixed points, a hard form of winner-take-all.

## 4 MINIMAX DYNAMICS

In the field of optimization, gradient descent-ascent is a standard method for finding saddle points of an objective function. This section of the paper explains the close relationship between gradient descent-ascent and excitatory-inhibitory networks[9, 10]. Furthermore, it reviews existing results on the convergence of gradient descent-ascent to saddle points[13, 10], which are the precedents of the convergence proofs of this paper.

The similarity of excitatory-inhibitory networks to gradient descent-ascent can be seen by comparing the partial derivatives of the saddle function (6) to the velocities $\dot{x}$ and $\dot{y}$,

$$-\frac{\partial S}{\partial x} = f^{-1}(\tau_x \dot{x} + x) - f^{-1}(x) \sim \tau_x \dot{x} , \tag{13}$$

$$\frac{\partial S}{\partial y} = g^{-1}(\tau_y \dot{y} + y) - g^{-1}(y) \sim \tau_y \dot{y} . \tag{14}$$

The notation $a \sim b$ means that the vectors $a$ and $b$ have the same signs, component by component. Because $f$ and $g$ are monotonic nondecreasing functions, $\dot{x}$ has the same signs as $-\partial S/\partial x$, while $\dot{y}$ has the same signs as $\partial S/\partial y$. In other words, the dynamics of the excitatory neurons tends to minimize $S$, while that of the inhibitory neurons tends to maximize $S$.

If the sign relation $\sim$ is replaced by equality in (13), we obtain a true gradient descent-ascent dynamics,

$$\tau_x \dot{x} = -\frac{\partial S}{\partial x} , \qquad \tau_y \dot{y} = \frac{\partial S}{\partial y} . \tag{15}$$

Sufficient conditions for convergence of gradient descent-ascent to saddle points are known[13, 10]. The conditions can be derived using a Lyapunov function constructed from the kinetic energy and the saddle function,

$$L = \frac{1}{2} \tau_x |\dot{x}|^2 + \frac{1}{2} \tau_y |\dot{y}|^2 + rS . \tag{16}$$

The time derivative of $L$ is given by

$$\dot{L} = -\dot{x}^T \frac{\partial^2 S}{\partial x^2} \dot{x} + \dot{y}^T \frac{\partial^2 S}{\partial y^2} \dot{y} - r\tau_x \dot{x}^2 + r\tau_y \dot{y}^2 \,. \tag{17}$$

Weak sufficient conditions can be derived with the choice $r = 0$, so that $L$ includes only kinetic energy terms. Then $L$ is obviously lower bounded by zero. Furthermore, $L$ is nonincreasing if $\partial^2 S/\partial x^2$ is positive definite for all $y$ and $\partial^2 S/\partial y^2$ is negative definite for all $x$. In this case, the existence of a unique saddle point is guaranteed, as $S$ is convex in $x$ for all $y$, and concave in $y$ for all $x$[13, 10].

If there is more than one saddle point, the kinetic energy by itself is generally not a Lyapunov function. This is because the dynamics may pass through the vicinity of more than one saddle point before it finally converges, so that the kinetic energy behaves nonmonotonically as a function of time. In this situation, some appropriate nonzero $r$ must be found.

The Lyapunov function (7) for excitatory-inhibitory networks is a generalization of the Lyapunov function (16) for gradient descent-ascent. This is analogous to the way in which the Lyapunov function for symmetric networks generalizes the potential function of gradient descent.

It should be noted that gradient descent-ascent is an unreliable way of finding a saddle point. It is easy to construct situations in which it leads to a limit cycle. The unreliability of gradient descent-ascent contrasts with the reliability of gradient descent at finding local minimum of a potential function. Similarly, symmetric networks converge to fixed points, but excitatory-inhibitory networks can converge to limit cycles as well.

## 5   HAMILTONIAN DYNAMICS

The dynamics of an excitatory-inhibitory network can be written in a dissipative Hamiltonian form. To do this, we define a phase space that is double the dimension of the state space, adding momenta $(p_x, p_y)$ that are canonically conjugate to $(x, y)$. The phase space dynamics

$$\tau_x \dot{x} + x = f(p_x) \,, \tag{18}$$

$$\tau_y \dot{y} + y = g(p_y) \,, \tag{19}$$

$$\left(r + \frac{d}{dt}\right)(u + Ax - By - p_x) = 0 \,, \tag{20}$$

$$\left(r + \frac{d}{dt}\right)(v + B^T x - Cy - p_y) = 0 \,, \tag{21}$$

reduces to the state space dynamics (1) on the affine space $A = \{(p_x, p_y, x, y) : p_x = u + Ax - By, p_y = v + B^T x - Cy\}$. Provided that $r > 0$, the affine space $A$ is an attractive invariant manifold.

Defining the Hamiltonian

$$H(p_x, x, p_y, y) = \tau_x^{-1} \Phi(p_x, x) + \tau_y^{-1} \Gamma(p_y, y) + rS(x, y) \,, \tag{22}$$

the phase space dynamics (18) can be written as

$$\dot{x} = \frac{\partial H}{\partial p_x} \,, \tag{23}$$

$$\dot{y} = \frac{\partial H}{\partial p_y} \,, \tag{24}$$

$$\dot{p_x} = -\frac{\partial H}{\partial x} + A\dot{x} - B\dot{y} - (\tau_x^{-1} + r)[p_x - f^{-1}(x)] , \qquad (25)$$

$$\dot{p_y} = -\frac{\partial H}{\partial y} + B^T\dot{x} - C\dot{y} - (\tau_y^{-1} - r)[p_y - g^{-1}(y)] \qquad (26)$$

$$+2r(v + B^T x - Cy - p_y) . \qquad (27)$$

On the invariant manifold $A$, the Hamiltonian is identical to the Lyapunov function (7) defined previously.

The rate of change of the energy is given by

$$
\begin{aligned}
\dot{H} = \ & \dot{x}^T A\dot{x} - (\tau_x^{-1} + r)\dot{x}^T[p_x - f^{-1}(x)] \qquad (28)\\
& -\dot{y}^T C\dot{y} - (\tau_y^{-1} - r)\dot{y}^T[p_y - g^{-1}(y)] \\
& +2r\dot{y}^T(v + B^T x - Cy - p_y) .
\end{aligned}
$$

The last term vanishes on the invariant manifold, leaving a result identical to (8). Therefore, if the noncanonical terms in the phase space dynamics (18) dissipate energy, then the Hamiltonian is nonincreasing. It is also possible that the velocity-dependent terms may pump energy into the system, rather than dissipate it, in which case oscillations or chaotic behavior may arise.

**Acknowledgments** This work was supported by Bell Laboratories. We would like to thank Eric Mjolsness for useful discussions.

# References

[1] M. A. Cohen and S. Grossberg. Absolute stability of global pattern formation and parallel memory storage by competitive neural networks. *IEEE*, 13:815–826, 1983.

[2] J. J. Hopfield. Neurons with graded response have collective computational properties like those of two-state neurons. *Proc. Natl. Acad. Sci. USA*, 81:3088–3092, 1984.

[3] J. J. Hopfield and D. W. Tank. Computing with neural circuits: a model. *Science*, 233:625–633, 1986.

[4] H. R. Wilson and J. D. Cowan. A mathematical theory of the functional dynamics of cortical and thalamic nervous tissue. *Kybernetik*, 13:55–80, 1973.

[5] Z. Li and J. J. Hopfield. Modeling the olfactory bulb and its neural oscillatory processings. *Biol. Cybern.*, 61:379–392, 1989.

[6] S. Amari. Dynamics of pattern formation in lateral-inhibition type neural fields. *Biol. Cybern.*, 27:77–87, 1977.

[7] B. Ermentrout. Complex dynamics in winner-take-all neural nets with slow inhibition. *Neural Networks*, 5:415–431, 1992.

[8] S. Amari and M. A. Arbib. Competition and cooperation in neural nets. In J. Metzler, editor, *Systems Neuroscience*, pages 119–165. Academic Press, New York, 1977.

[9] E. Mjolsness and C. Garrett. Algebraic transformations of objective functions. *Neural Networks*, 3:651–669, 1990.

[10] J. C. Platt and A. H. Barr. Constrained differential optimization. In D. Z. Anderson, editor, *Neural Information Processing Systems*, page 55, New York, 1987. American Institute of Physics.

[11] I. M. Elfadel. Convex potentials and their conjugates in analog mean-field optimization. *Neural Computation*, 7(5):1079–1104, 1995.

[12] J. D. Cowan. A statistical mechanics of nervous activity. In *Some mathematical questions in biology*, volume III. AMS, 1972.

[13] K. J. Arrow, L. Hurwicz, and H. Uzawa. *Studies in linear and non-linear programming*. Stanford University, Stanford, 1958.
